# On the Completeness of First-Order Knowledge Compilation for Lifted Probabilistic Inference

**Guy Van den Broeck**
Department of Computer Science, Katholieke Universiteit Leuven
Celestijnenlaan 200A, B-3001 Heverlee, Belgium
`guy.vandenbroeck@cs.kuleuven.be`

## Abstract

Probabilistic logics are receiving a lot of attention today because of their expressive power for knowledge representation and learning. However, this expressivity is detrimental to the tractability of inference, when done at the propositional level. To solve this problem, various lifted inference algorithms have been proposed that reason at the first-order level, about groups of objects as a whole. Despite the existence of various lifted inference approaches, there are currently no completeness results about these algorithms. The key contribution of this paper is that we introduce a formal definition of lifted inference that allows us to reason about the completeness of lifted inference algorithms relative to a particular class of probabilistic models. We then show how to obtain a completeness result using a first-order knowledge compilation approach for theories of formulae containing up to two logical variables.

## 1 Introduction and related work

Probabilistic logic models build on first-order logic to capture relational structure and on graphical models to represent and reason about uncertainty [1, 2]. Due to their expressivity, these models can concisely represent large problems with many interacting random variables. While the semantics of these logics is often defined through grounding the models [3], performing inference at the propositional level is – as for first-order logic – inefficient. This has motivated the quest for lifted inference methods that exploit the structure of probabilistic logic models for efficient inference, by reasoning about groups of objects as a whole and avoiding repeated computations. The first approaches to exact lifted inference have upgraded the variable elimination algorithm to the first-order level [4, 5, 6]. More recent work is based on methods from logical inference [7, 8, 9, 10], such as knowledge compilation. While these approaches often yield dramatic improvements in runtime over propositional inference methods on specific problems, it is still largely unclear for which classes of models these lifted inference operators will be useful and for which ones they will eventually have to resort to propositional inference. One notable exception in this regard is lifted belief propagation [11], which performs exact lifted inference on any model whose factor graph representation is a tree.

A first contribution of this paper is that we introduce a notion of domain lifted inference, which formally defines what lifting means, and which can be used to characterize the classes of probabilistic models to which lifted inference applies. Domain lifted inference essentially requires that probabilistic inference runs in polynomial time in the domain size of the logical variables appearing in the model. As a second contribution we show that the class of models expressed as 2-WFOMC formulae (weighted first-order model counting with up to 2 logical variables per formula) can be domain lifted using an extended first-order knowledge compilation approach [10]. The resulting approach allows for lifted inference even in the presence of (anti-) symmetric or total relations in a theory. These are extremely common and useful concepts that cannot be lifted by any of the existing first-order knowledge compilation inference rules.

## 2  Background

We will use standard concepts of function-free first-order logic (FOL). An *atom* $p(t_1, \ldots, t_n)$ consists of a predicate $p/n$ of arity $n$ followed by $n$ arguments, which are either *constants* or *logical variables*. An atom is *ground* if it does not contain any variables. A *literal* is an atom $a$ or its negation $\neg a$. A *clause* is a disjunction $l_1 \vee \ldots \vee l_k$ of literals. If $k = 1$, it is a *unit* clause. An *expression* is an atom, literal or clause. The $\mathrm{pred}(a)$ function maps an atom to its predicate and the $\mathrm{vars}(e)$ function maps an expression to its logical variables. A theory in conjunctive normal form (CNF) is a conjunction of clauses. We often represent theories by their set of clauses and clauses by their set of literals. Furthermore, we will assume that all logical variables are universally quantified.

In addition, we associate a set of constraints with each clause or atom, either of the form $X \neq t$, where $X$ is a logical variable and $t$ is a constant or variable, or of the form $X \in D$, where $D$ is a domain, or the negation of these constraints. These define a finite domain for each logical variable. Abusing notation, we will use constraints of the form $X = t$ to denote a substitution of $X$ by $t$. The function $\mathrm{atom}(e)$ maps an expression $e$ to its atoms, now associating the constraints on $e$ with each atom individually. To add the constraint $c$ to an expression $e$, we use the notation $e \wedge c$. Two atoms *unify* if there is a substitution which makes them identical and if the conjunction of the constraints on both atoms with the substitution is satisfiable. Two expressions $e_1$ and $e_2$ are *independent*, written $e_1 \perp\!\!\!\perp e_2$, if no atom $a_1 \in \mathrm{atom}(e_1)$ unifies with an atom $a_2 \in \mathrm{atom}(e_2)$.

We adopt the Weighted First-Order Model Counting (WFOMC) [10] formalism to represent probabilistic logic models, building on the notion of a Herbrand interpretation. Herbrand interpretations are subsets of the Herbrand base $HB(T)$, which consists of all ground atoms that can be constructed with the available predicates and constant symbols in $T$. The atoms in a Herbrand interpretation are assumed to be true. All other atoms in $HB(T)$ are assumed to be false. An interpretation $I$ satisfies a theory $T$, written as $I \models T$, if it satisfies all the clauses $c \in T$. The WFOMC problem is defined on a weighted logic theory $T$, which is a logic theory augmented with a positive weight function $\mathrm{w}$ and a negative weight function $\overline{\mathrm{w}}$, which assign a weight to each predicate. The WFOMC problem involves computing

$$\mathrm{wmc}(T, \mathrm{w}, \overline{\mathrm{w}}) = \sum_{I \models T} \prod_{a \in I} \mathrm{w}(\mathrm{pred}(a)) \prod_{a \in HB(T) \setminus I} \overline{\mathrm{w}}(\mathrm{pred}(a)). \tag{1}$$

## 3  First-order knowledge compilation for lifted probabilistic inference

### 3.1  Lifted probabilistic inference

A first-order probabilistic model defines a probability distribution $\mathrm{P}$ over the set of Herbrand interpretations $H$. Probabilistic inference in these models is concerned with computing the posterior probability $\mathrm{P}(q|e)$ of query $q$ given evidence $e$, where $q$ and $e$ are logical expressions in general:

$$\mathrm{P}(q|e) = \frac{\sum_{h \in H, h \models q \wedge e} \mathrm{P}(h)}{\sum_{h \in H, h \models e} \mathrm{P}(h)} \tag{2}$$

We propose one notion of lifted inference for first-order probabilistic models, defined in terms of the computational complexity of inference w.r.t. the domains of logical variables. It is clear that other notions of lifted inference are conceivable, especially in the case of approximate inference.

**Definition 1** (Domain Lifted Probabilistic Inference). A probabilistic inference procedure is *domain lifted* for a model $m$, query $q$ and evidence $e$ iff the inference procedure runs in *polynomial* time in $|D_1|, \ldots, |D_k|$ with $D_i$ the domain of the logical variable $v_i \in \mathrm{vars}(m, q, e)$.

Domain lifted inference does not prohibit the algorithm to be *exponential* in the size of the vocabulary, that is, the number of predicates, arguments and constants, of the probabilistic model, query and evidence. In fact, the definition allows inference to be exponential in the number of constants which occur in arguments of atoms in the theory, query or evidence, as long as it is *polynomial* in the cardinality of the logical variable domains. This definition of lifted inference stresses the ability to efficiently deal with the domains of the logical variables that arise, regardless of their size, and formalizes what seems to be generally accepted in the lifted inference literature.

A class of probabilistic models is a set of probabilistic models expressed in a particular formalism. As examples, consider Markov logic networks (MLN) [12] or parfactors [4], or the weighted FOL theories for WFOMC that we introduced above, when the weights are normalized.

**Definition 2** (Completeness). Restricting queries to atoms and evidence to a conjunction of literals, a procedure that is domain lifted for all probabilistic models $m$ in a class of models $M$ and for all queries $q$ and evidence $e$, is called *complete* for $M$.

### 3.2 First-order knowledge compilation

First-order knowledge compilation is an approach to lifted probabilistic inference consisting of the following three steps (see Van den Broeck et al. [10] for details):

1. Convert the probabilistic logical model to a weighted CNF. Converting MLNs or parfactors requires adding new atoms to the theory that represent the (truth) value of each factor or formula.

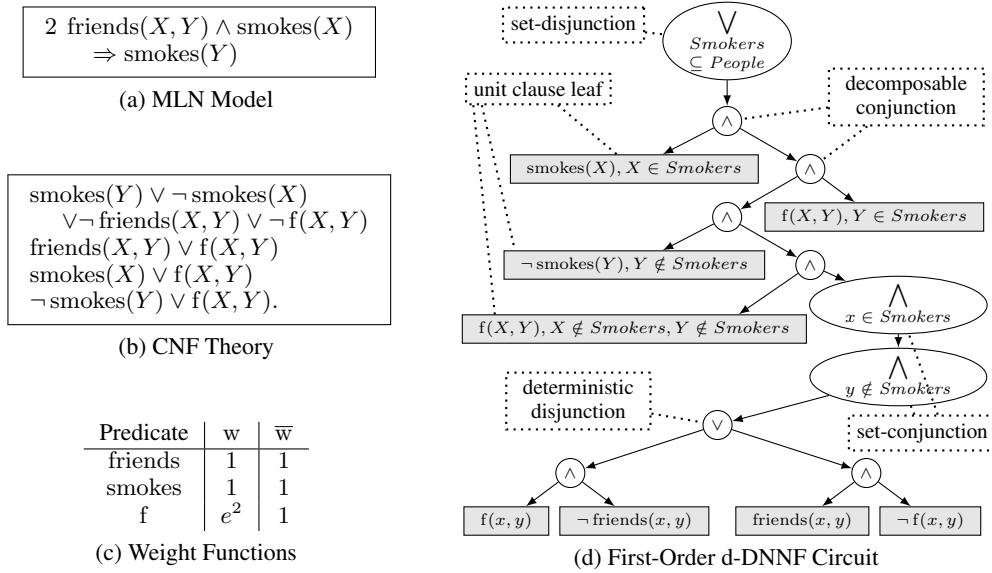

(a) MLN Model

(b) CNF Theory

(c) Weight Functions

(d) First-Order d-DNNF Circuit

Figure 1: Friends-smokers example (taken from [10])

**Example 1.** The MLN in Figure 1a assigns a weight to a formula in FOL. Figure 1b represents the same model as a weighted CNF, introducing a new atom $f(X, Y)$ to encode the truth value of the MLN formula. The probabilistic information is captured by the weight functions in Figure 1c.

2. Compile the logical theory into a First-Order d-DNNF (FO d-DNNF) circuit. Figure 1d shows an example of such a circuit. Leaves represent unit clauses. Inner nodes represent the disjunction or conjunction of their children $l$ and $r$, but with the constraint that disjunctions must be deterministic ($l \wedge r$ is unsatisfiable) and conjunctions must be decomposable ($l \perp\!\!\!\perp r$).

3. Perform WFOMC inference to compute posterior probabilities. In a FO d-DNNF circuit, WFOMC is polynomial in the size of the circuit and the cardinality of the domains.

To compile the CNF theory into a FO d-DNNF circuit, Van den Broeck et al. [10] propose a set of compilation rules, which we will refer to as $CR_1$. We will now briefly describe these rules.

*Unit Propagation* introduces a decomposable conjunction when the theory contains a unit clause. *Independence* creates a decomposable conjunction when the theory contains independent subtheories. *Shannon decomposition* applies when the theory contains ground atoms and introduces a deterministic disjunction between two modified theories: one where the ground atom is true, and one where it is false. *Shattering* splits clauses in the theory until all pairs of atoms represent either a disjoint or identical set of ground atoms.

**Example 2.** In Figure 2a, the first two clauses are made independent from the $\text{friends}(X, X)$ clause and split off in a decomposable conjunction by unit propagation. The unit clause becomes a leaf of the FO d-DNNF circuit, while the other operand requires further compilation.

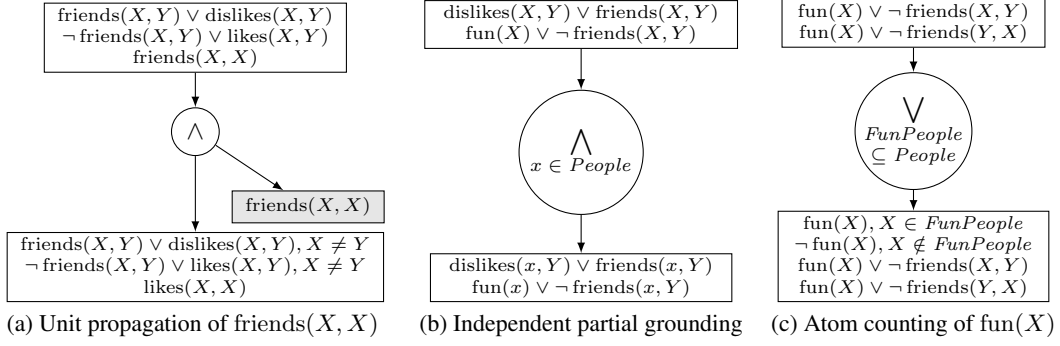

(a) Unit propagation of friends$(X, X)$  (b) Independent partial grounding  (c) Atom counting of fun$(X)$

Figure 2: Examples of compilation rules. Circles are FO d-DNNF inner nodes. White rectangles show theories before and after applying the rule. All variable domains are $People$. (taken from [10])

*Independent Partial Grounding* creates a decomposable conjunction over a set of child circuits, which are identical up to the value of a grounding constant. Since they are structurally identical, only one child circuit is actually compiled. *Atom Counting* applies when the theory contains an atom with a single logical variable $X \in D$. It explicitly represents the domain $D^\top \subseteq D$ of $X$ for which the atom is true. It compiles the theory into a deterministic disjunction between all possible such domains. Again, these child circuits are identical up to the value of $D^\top$ and only one is compiled.

**Example 3.** The theory in Figure 2b is compiled into a decomposable set-conjunction of theories that are independent and identical up to the value of the $x$ constant. The theory in Figure 2c contains an atom with one logical variable: fun$(X)$. Atom counting compiles it into a deterministic set-disjunction over theories that differ in $FunPeople$, which is the domain of $X$ for which fun$(X)$ is true. Subsequent steps of unit propagation remove the fun$(X)$ atoms from the theory entirely.

## 3.3 Completeness

We will now characterize those theories where the $CR_1$ compilation rules cannot be used, and where the inference procedure has to resort to grounding out the theory to propositional logic. For these, first-order knowledge compilation using $CR_1$ is not yet domain lifted.

When a logical theory contains symmetric, anti-symmetric or total relations, such as

$$\text{friends}(X, Y) \Rightarrow \text{friends}(Y, X), \tag{3}$$

$$\text{parent}(X, Y) \Rightarrow \neg \text{parent}(Y, X), X \neq Y, \tag{4}$$

$$\leq (\text{X}, \text{Y}) \vee \leq (\text{Y}, \text{X}), \tag{5}$$

or more general formulas, such as

$$\text{enemies}(X, Y) \Rightarrow \neg \text{friend}(X, Y) \wedge \neg \text{friend}(Y, X), \tag{6}$$

none of the $CR_1$ rules apply. Intuitively, the underlying problem is the presence of either:

- Two unifying (not independent) atoms in the same clause which contain the same logical variable in different positions of the argument list. Examples include (the CNF of) Formulas 3, 4 and 5, where the $X$ and $Y$ variable are bound by unifying two atoms from the same clause.

- Two logical variables that bind when unifying one pair of atoms but appear in different positions of the argument list of two other unifying atoms. Examples include Formula 6, which in CNF is

$$\neg \text{friend}(X, Y) \vee \neg \text{enemies}(X, Y)$$
$$\neg \text{friend}(Y, X) \vee \neg \text{enemies}(X, Y)$$

Here, unifying the enemies$(X, Y)$ atoms binds the $X$ variables from both clauses, which appear in different positions of the argument lists of the unifying atoms friend$(X, Y)$ and friend$(Y, X)$.

Both of these properties preclude the use of $CR_1$ rules. Also in the context of other model classes, such as MLNs, probabilistic versions of the above formulas cannot be processed by $CR_1$ rules.

Even though first-order knowledge compilation with $CR_1$ rules does not have a clear completeness result, we can show some properties of theories to which none of the compilation rules apply. First, we need to distinguish between the arity of an atom and its dimension. A predicate with arity two might have atoms with dimension one, when one of the arguments is ground or both are identical.

**Definition 3** (Dimension of an Expression)**.** The dimension of an expression $e$ is the number of logical variables it contains: $\dim(e) = |\operatorname{vars}(e)|$.

**Lemma 1** ($CR_1$ Postconditions)**.** *The $CR_1$ rules remove all atoms from the theory $T$ which have zero or one logical variable arguments, such that afterwards $\forall a \in \operatorname{atom}(T) : \dim(a) > 1$. When no $CR_1$ rule applies, the theory is shattered and contains no independent subtheories.*

*Proof.* Ground atoms are removed by the Shannon decomposition operator followed by unit propagation. Atoms with a single logical variable (including unary relations) are removed by the atom counting operator followed by unit propagation. If $T$ contains independent subtheories, the independence operator can be applied. Shattering is always applied when $T$ is not yet shattered. $\square$

## 4 Extending first-order knowledge compilation

In this section we introduce a new operator which does apply to the theories from Section 3.3.

### 4.1 Logical variable properties

To formally define the operator we propose, and prove its correctness, we first introduce some mathematical concepts related to the logical variables in a theory (partly after Jha et al. [8]).

**Definition 4** (Binding Variables)**.** Two logical variables $X, Y$ are *directly binding* $\operatorname{b}(X, Y)$ if they are bound by unifying a pair of atoms in the theory. The *binding* relationship $\operatorname{b}^+(X, Y)$ is the transitive closure of the directly binding relation $\operatorname{b}(X, Y)$.

**Example 4.** In the theory
$$\neg \operatorname{p}(W, X) \vee \neg \operatorname{q}(X)$$
$$\operatorname{r}(Y) \vee \neg \operatorname{q}(Y)$$
$$\neg \operatorname{r}(Z) \vee \operatorname{s}(Z)$$

the variable pairs $(X, Y)$ and $(Y, Z)$ are directly binding. The variables $X, Y$ and $Z$ are binding. Variable $W$ does not bind to any other variable. Note that the binding relationship $\operatorname{b}^+(X, Y)$ is an equivalence relation that defines two equivalence classes: $\{X, Y, Z\}$ and $\{W\}$.

**Lemma 2** (Binding Domains)**.** *After shattering, binding logical variables have identical domains.*

*Proof.* During shattering (see Section 3.2), when two atoms unify, binding two variables with partially overlapping domains, the atoms' clauses are split up into clauses where the domain of the variables is identical, and clauses where the domains are disjoint and the atoms no longer unify. $\square$

**Definition 5** (Root Binding Class)**.** A *root variable* is a variable that appears in all the atoms in its clause. A *root binding class* is an equivalence class of binding variables where all variables are root.

**Example 5.** In the theory of Example 4, $\{X, Y, Z\}$ is a root binding class and $\{W\}$ is not.

### 4.2 Domain recursion

We will now introduce the new domain recursion operator, starting with its preconditions.

**Definition 6.** A theory allows for domain recursion when (i) the theory is shattered, (ii) the theory contains no independent subtheories and (iii) there exists a root binding class.

From now on, we will denote with $C$ the set of clauses of the theory at hand and with $B$ a root binding class guaranteed to exist if $C$ allows for domain recursion. Lemma 2 states that all variables in $B$ have identical domains. We will denote the domain of these variables with $D$.

The intuition behind the domain recursion operator is that it modifies $D$ by making one element explicit: $D = D' \cup \{x_D\}$ with $x_D \notin D'$. This explicit domain element is introduced by the SPLITD function, which splits clauses w.r.t. the new subdomain $D'$ and element $x_D$.

**Definition 7** (SPLITD). For a clause $c$ and given set of variables $\mathcal{V}_c \subseteq \text{vars}(c)$ with domain $D$, let

$$\text{SPLITD}(c, \mathcal{V}_c) = \begin{cases} c, & \text{if } \mathcal{V}_c = \emptyset \\ \text{SPLITD}(c_1, \mathcal{V}_c \setminus \{V\}) \cup \text{SPLITD}(c_2, \mathcal{V}_c \setminus \{V\}), & \text{if } \mathcal{V}_c \neq \emptyset \end{cases} \quad (7)$$

where $c_1 = c \wedge (V = x_D)$ and $c_2 = c \wedge (V \neq x_D) \wedge (V \in D')$ for some $V \in \mathcal{V}_c$. For a set of clauses $C$ and set of variables $\mathcal{V}$ with domain $D$: $\text{SPLITD}(C, \mathcal{V}) = \bigcup_{c \in C} \text{SPLITD}(c, \mathcal{V} \cap \text{vars}(c))$.

The domain recursion operator creates three sets of clauses: $\text{SPLITD}(C, B) = C_x \cup C_v \cup C_r$, with

$$C_x = \{c \wedge \bigwedge_{V \in B \cap \text{vars}(c)} (V = x_D) | c \in C\}, \quad (8)$$

$$C_v = \{c \wedge \bigwedge_{V \in B \cap \text{vars}(c)} (V \neq x_D) \wedge (V \in D') | c \in C\}, \quad (9)$$

$$C_r = \text{SPLITD}(C, B) \setminus C_x \setminus C_v. \quad (10)$$

**Proposition 3.** *The conjunction of the domain recursion sets is equivalent to the original theory:* $\bigwedge_{c \in C} c \equiv \bigwedge_{c \in \text{SPLITD}(C,B)} c$ *and therefore* $\bigwedge_{c \in C} c \equiv \left(\bigwedge_{c \in C_x} c\right) \wedge \left(\bigwedge_{c \in C_v} c\right) \wedge \left(\bigwedge_{c \in C_r} c\right)$.

We will now show that these sets are independent and that their conjunction is decomposable.

**Theorem 4.** *The theories $C_x$, $C_v$ and $C_r$ are independent: $C_x \perp\!\!\!\perp C_v$, $C_x \perp\!\!\!\perp C_r$ and $C_v \perp\!\!\!\perp C_r$.*

The proof of Theorem 4 relies on the following Lemma.

**Lemma 5.** *If the theory allows for domain recursion, all clauses and atoms contain the same number of variables from $B$:*

$$\exists n, \forall c \in C, \forall a \in \text{atom}(C) : |\text{vars}(c) \cap B| = |\text{vars}(a) \cap B| = n.$$

*Proof.* Denote with $C_n$ the clauses in $C$ that contain $n$ logical variables from $B$ and with $C_n^c$ its compliment in $C$. If $C$ is nonempty, there is a $n > 0$ for which $C_n$ is nonempty. Then every atom in $C_n$ contains exactly $n$ variables from $B$ (Definition 5). Since the theory contains no independent subtheories, there must be an atom $a$ in $C_n$ which unifies with an atom $a_c$ in $C_n^c$, or $C_n^c$ is empty. After shattering, all unifications bind one variable from $a$ to a single variable from $a_c$. Because $a$ contains exactly $n$ variables from $B$, $a_c$ must also contain exactly $n$ (Definition 4), and because $B$ is a root binding class, the clause of $a_c$ also contains exactly $n$, which contradicts the definition of $C_n^c$. Therefore, $C_n^c$ is empty, and because the variables in $B$ are root, they also appear in all atoms. $\square$

*Proof of Theorem 4.* From Lemma 5, all atoms in $C$ contain the same number of variables from $B$. In $C_x$, these variables are all constrained to be equal to $x_D$, while in $C_v$ and $C_r$ at least one variable is constrained to be different from $x_D$. An attempt to unify an atom from $C_x$ with an atom from $C_v$ or $C_r$ therefore creates an unsatisfiable set of constraints. Similarly, atoms from $C_v$ and $C_r$ cannot be unified. $\square$

Finally, we extend the FO d-DNNF language proposed in Van den Broeck et al. [10] with a new node, the recursive decomposable conjunction $\bigwedge_r$, and define the domain recursion compilation rule.

**Definition 8** ($\bigwedge_r$). The FO d-DNNF node $\bigwedge_r(n_x, n_r, D, D', \mathcal{V})$ represents a decomposable conjunction between the d-DNNF nodes $n_x$, $n_r$ and a d-DNNF node isomorphic to the $\bigwedge_r$ node itself. In particular, the isomorphic operand is identical to the node itself, except for the size of the domain of the variables in $\mathcal{V}$, which becomes one smaller, going from $D$ to $D'$ in the isomorphic operand.

We have shown that the conjunction between sets $C_x$, $C_v$ and $C_r$ is decomposable (Theorem 4) and logically equivalent to the original theory (Proposition 3). Furthermore, $C_v$ is identical to $C$, up to the constraints on the domain of the variables in $B$. This leads us to the following definition of domain recursion.

**Definition 9** (Domain Recursion). The domain recursion compilation rule compiles $C$ into $\bigwedge_r(n_x, n_r, D, D', B)$, where $n_x, n_r$ are the compiled circuits for $C_x, C_r$. The third set $C_v$ is represented by the recursion on $D$, according to Definition 8.

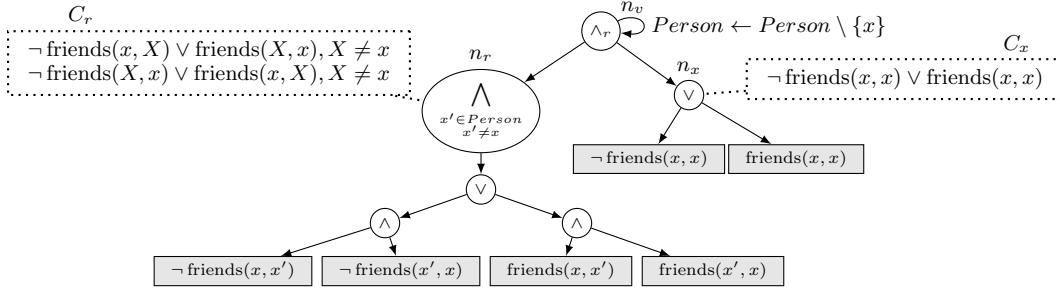

Figure 3: Circuit for the symmetric relation in Equation 3, rooted in a recursive conjunction.

**Example 6.** Figure 3 shows the FO d-DNNF circuit for Equation 3. The theory is split up into three independent theories: $C_r$ and $C_x$, shown in the Figure 3, and $C_v = \{\neg\,\mathrm{friends}(X,Y) \vee$ $\mathrm{friends}(Y,X), X \neq x, Y \neq x\}$. The conjunction of these theories is equivalent to Equation 3. Theory $C_v$ is identical to Equation 3, up to the inequality constraints on $X$ and $Y$.

**Theorem 6.** *Given a function* size*, which maps domains to their size, the weighted first-order model count of a $\bigwedge_r(n_x, n_r, D, D', \mathcal{V})$ node is*

$$\mathrm{wmc}(\bigwedge_r(n_x, n_r, D, D', \mathcal{V}), \mathrm{size}) = \mathrm{wmc}(n_x, \mathrm{size})^{\mathrm{size}(D)} \prod_{s=0}^{\mathrm{size}(D)} \mathrm{wmc}(n_r, \mathrm{size} \cup \{D' \mapsto s\}),$$

(11)

*where* $\mathrm{size} \cup \{D' \mapsto s\}$ *adds to the* size *function that the subdomain $D'$ has cardinality s.*

*Proof.* If $C$ allows for domain recursion, due to Theorem 4, the weighted model count is

$$\mathrm{wmc}(C, \mathrm{size}) = \begin{cases} 1, & \text{if } \mathrm{size}(D) = 0 \\ \mathrm{wmc}(C_x) \cdot \mathrm{wmc}(C_v, \mathrm{size}') \cdot \mathrm{wmc}(C_r, \mathrm{size}') & \text{if } \mathrm{size}(D) > 0 \end{cases}$$

(12)

where $\mathrm{size}' = \mathrm{size} \cup \{D' \mapsto \mathrm{size}(D) - 1\}$. $\square$

**Theorem 7.** *The Independent Partial Grounding compilation rule is a special case of the domain recursion rule, where $\forall c \in C : |\,\mathrm{vars}(c) \cap B\,| = 1$ (and therefore $C_r = \emptyset$).*

### 4.3 Completeness

In this section, we introduce a class of models for which first-order knowledge compilation with domain recursion is complete.

**Definition 10** ($k$-WFOMC). The class of $k$-WFOMC consist of WFOMC theories with clauses that have up to $k$ logical variables.

A first completeness result is for 2-WFOMC, using the set of knowledge compilation rules $CR_2$, which are the rules in $CR_1$ extended with domain recursion.

**Theorem 8** (Completeness for 2-WFOMC). *First-order knowledge compilation using the $CR_2$ compilation rules is a complete domain lifted probabilistic inference algorithm for 2-WFOMC.*

*Proof.* From Lemma 1, after applying the $CR_1$ rules, the theory contains only atoms with dimension larger than or equal to two. From Definition 10, each clause has dimension smaller than or equal to two. Therefore, each logical variable in the theory is a root variable and according to Definition 5, every equivalence class of binding variables is a root binding class. Because of Lemma 1, the theory allows for domain recursion, which requires further compilation of two theories: $C_x$ and $C_r$ into $n_x$ and $n_r$. Both have dimension smaller than 2 and can be lifted by $CR_1$ compilation rules. $\square$

The properties of 2-WFOMC are a sufficient but not necessary condition for first-order knowledge compilation to be domain lifted. We can obtain a similar result for MLNs or parfactors by reducing them to a WFOMC problem. If an MLN contains only formulae with up to $k$ logical variables, then its WFOMC representation will be in $k$-WFOMC.

This result for 2-WFOMC is not trivial. Van den Broeck et al. [10] showed in their experiments that counting first-order variable elimination (C-FOVE) [6] fails to lift the "Friends Smoker Drinker" problem, which is in 2-WFOMC. We will show in the next section that the $CR_1$ rules fail to lift the theory in Figure 4a, which is in 2-WFOMC. Note that there are also useful theories that are not in 2-WFOMC, such as those containing the transitive relation $\text{friends}(X, Y) \land \text{friends}(Y, Z) \Rightarrow \text{friends}(X, Z)$.

## 5 Empirical evaluation

To complement the theoretical results of the previous section, we extended the WFOMC implementation[1] with the domain recursion rule. We performed experiments with the theory in Figure 4a, which is a version of the *friends and smokers* model [11] extended with the symmetric relation of Equation 3. We evaluate the performance querying $\text{P}(\text{smokes}(bob))$ with increasing domain size, comparing our approach to the existing WFOMC implementation and its propositional counterpart, which first grounds the theory and then compiles it with the c2d compiler [13] to a propositional d-DNNF circuit. We did not compare to C-FOVE [6] because it cannot perform lifted inference on this model.

Propositional inference quickly becomes intractable when there are more than 20 people. The lifted inference algorithms scale much better. The $CR_1$ rules can exploit some regularities in the model. For example, they eliminate all the $\text{smokes}(X)$ atoms from the theory. They do, however, resort to grounding at a later stage of the compilation process. With the domain recursion rule, there is no need for grounding. This advantage is clear in the experiments, our approach having an almost constant inference time in this range of domains sizes. Note that the runtimes for c2d include compilation and evaluation of the circuit, whereas the WFOMC runtimes only represent evaluation of the FO d-DNNF. After all, propositional compilation depends on the domain size but first-order compilation does not. First-order compilation takes a constant two seconds for both rule sets.

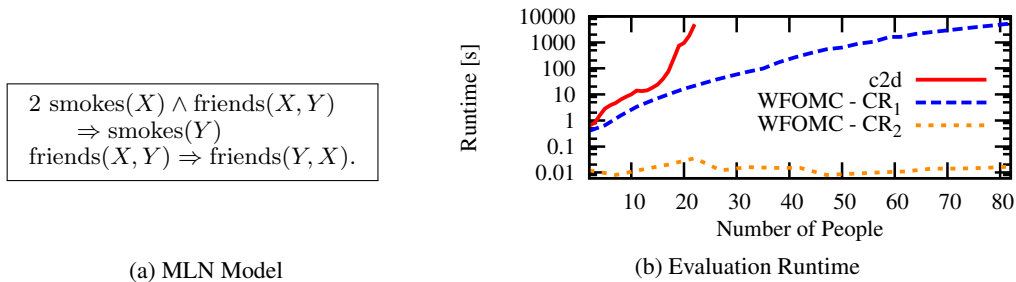

|  |
| --- |
| $2\ \text{smokes}(X) \land \text{friends}(X, Y)$ |
| $\Rightarrow \text{smokes}(Y)$ |
| $\text{friends}(X, Y) \Rightarrow \text{friends}(Y, X).$ |

(a) MLN Model            (b) Evaluation Runtime

Figure 4: Symmetric friends and smokers experiment, comparing propositional knowledge compilation (c2d) to WFOMC using compilation rules $CR_1$ and $CR_2$ (which includes domain recursion).

## 6 Conclusions

We proposed a definition of complete domain lifted probabilistic inference w.r.t. classes of probabilistic logic models. This definition considers algorithms to be lifted if they are polynomial in the size of logical variable domains. Existing first-order knowledge compilation turns out not to admit an intuitive completeness result. Therefore, we generalized the existing Independent Partial Grounding compilation rule to the domain recursion rule. With this one extra rule, we showed that first-order knowledge compilation is complete for a significant class of probabilistic logic models, where the WFOMC representation has up to two logical variables per clause.

### Acknowledgments

The author would like to thank Luc De Raedt, Jesse Davis and the anonymous reviewers for valuable feedback. This work was supported by the Research Foundation-Flanders (FWO-Vlaanderen).

## Footnotes

[1]http://dtai.cs.kuleuven.be/wfomc/

# References

[1] Lise Getoor and Ben Taskar, editors. *An Introduction to Statistical Relational Learning*. MIT Press, 2007.

[2] Luc De Raedt, Paolo Frasconi, Kristian Kersting, and Stephen Muggleton, editors. *Probabilistic inductive logic programming: theory and applications*. Springer-Verlag, Berlin, Heidelberg, 2008.

[3] Daan Fierens, Guy Van den Broeck, Ingo Thon, Bernd Gutmann, and Luc De Raedt. Inference in probabilistic logic programs using weighted CNF's. In *Proceedings of UAI*, pages 256–265, 2011.

[4] David Poole. First-order probabilistic inference. In *Proceedings of IJCAI*, pages 985–991, 2003.

[5] Rodrigo de Salvo Braz, Eyal Amir, and Dan Roth. Lifted first-order probabilistic inference. In *Proceedings of IJCAI*, pages 1319–1325, 2005.

[6] Brian Milch, Luke S. Zettlemoyer, Kristian Kersting, Michael Haimes, and Leslie Pack Kaelbling. Lifted Probabilistic Inference with Counting Formulas. In *Proceedings of AAAI*, pages 1062–1068, 2008.

[7] Vibhav Gogate and Pedro Domingos. Exploiting Logical Structure in Lifted Probabilistic Inference. In *Proceedings of StarAI*, 2010.

[8] Abhay Jha, Vibhav Gogate, Alexandra Meliou, and Dan Suciu. Lifted Inference Seen from the Other Side: The Tractable Features. In *Proceedings of NIPS*, 2010.

[9] Vibhav Gogate and Pedro Domingos. Probabilistic theorem proving. In *Proceedings of UAI*, pages 256–265, 2011.

[10] Guy Van den Broeck, Nima Taghipour, Wannes Meert, Jesse Davis, and Luc De Raedt. Lifted Probabilistic Inference by First-Order Knowledge Compilation. In *Proceedings of IJCAI*, pages 2178–2185, 2011.

[11] Parag Singla and Pedro Domingos. Lifted first-order belief propagation. In *Proceedings of AAAI*, pages 1094–1099, 2008.

[12] Matthew Richardson and Pedro Domingos. Markov logic networks. *Machine Learning*, 62(1): 107–136, 2006.

[13] Adnan Darwiche. New advances in compiling CNF to decomposable negation normal form. In *Proceedings of ECAI*, pages 328–332, 2004.

